# Variational inference for Markov jump processes

**Manfred Opper**
Department of Computer Science
Technische Universität Berlin
D-10587 Berlin, Germany
opperm@cs.tu-berlin.de

**Guido Sanguinetti**
Department of Computer Science
University of Sheffield, U.K.
guido@dcs.shef.ac.uk

## Abstract

Markov jump processes play an important role in a large number of application domains. However, realistic systems are analytically intractable and they have traditionally been analysed using simulation based techniques, which do not provide a framework for statistical inference. We propose a mean field approximation to perform posterior inference and parameter estimation. The approximation allows a practical solution to the inference problem, while still retaining a good degree of accuracy. We illustrate our approach on two biologically motivated systems.

## Introduction

Markov jump processes (MJPs) underpin our understanding of many important systems in science and technology. They provide a rigorous probabilistic framework to model the joint dynamics of groups (*species*) of interacting individuals, with applications ranging from information packets in a telecommunications network to epidemiology and population levels in the environment. These processes are usually non-linear and highly coupled, giving rise to non-trivial steady states (often referred to as emerging properties). Unfortunately, this also means that exact statistical inference is unfeasible and approximations must be made in the analysis of these systems.

A traditional approach, which has been very successful throughout the past century, is to ignore the discrete nature of the processes and to approximate the stochastic process with a deterministic process whose behaviour is described by a system of non-linear, coupled ODEs. This approximation relies on the stochastic fluctuations being negligible compared to the average population counts. There are many important situations where this assumption is untenable: for example, stochastic fluctuations are reputed to be responsible for a number of important biological phenomena, from cell differentiation to pathogen virulence [1]. Researchers are now able to obtain accurate estimates of the number of macromolecules of a certain species within a cell [2, 3], prompting a need for practical statistical tools to handle discrete data.

Sampling approaches have been extensively used to simulate the behaviour of MJPs. Gillespie's algorithm and its generalisations [4, 5] form the basis of many simulators used in systems biology studies. The simulations can be viewed as individual samples taken from a completely specified MJP, and can be very useful to reveal possible steady states. However, it is not clear how observed data can be incorporated in a principled way, which renders this approach of limited use for posterior inference and parameter estimation. A Markov chain Monte Carlo (MCMC) approach to incorporate observations has been recently proposed by Boys *et al.* [6]. While this approach holds a lot of promise, it is computationally very intensive. Despite several simplifying approximations, the correlations between samples mean that several millions of MCMC iterations are needed even in simple examples. In this paper we present an alternative, deterministic approach to posterior inference and parameter estimation in MJPs. We extend the mean-field (MF) variational approach ([*cf. e.g.* 7]) to approximate a probability distribution over an (infinite dimensional) space of discrete *paths*, representing the time-evolving state of the system. In this way, we replace the couplings between the

different species by their average, mean-field (MF) effect. The result is an iterative algorithm that allows parameter estimation and prediction with reasonable accuracy and very contained computational costs.

The rest of this paper is organised as follows: in sections 1 and 2 we review the theory of Markov jump processes and introduce our general strategy to obtain a MF approximation. In section 3 we introduce the Lotka-Volterra model which we use as an example to describe how our approach works. In section 4 we present experimental results on simulated data from the Lotka-Volterra model and from a simple gene regulatory network. Finally, we discuss the relationship of our study to other stochastic models, as well as further extensions and developments of our approach.

# 1 Markov jump processes

We start off by establishing some notation and basic definitions. A $D$-dimensional *discrete stochastic process* is a family of $D$-dimensional discrete random variables $\mathbf{x}(t)$ indexed by the continuous time $t$. In our examples, the values taken by $\mathbf{x}(t)$ will be restricted to the non-negative integers $\mathbb{N}_0^D$. The dimensionality $D$ represents the number of (molecular) species present in the system; the components of the vector $\mathbf{x}(t)$ then represent the number of individuals of each species present at time $t$. Furthermore, the stochastic processes we will consider will always be Markovian, *i.e.* given any sequence of observations for the state of the system $(\mathbf{x}_{t_1}, \ldots, \mathbf{x}_{t_N})$, the conditional probability of the state of the system at a subsequent time $\mathbf{x}_{t_{N+1}}$ depends only on the last of the previous observations. A discrete stochastic process which exhibits the Markov property is called a *Markov jump process* (MJP).

A MJP is characterised by its *process rates* $f(\mathbf{x}'|\mathbf{x})$, defined $\forall \mathbf{x}' \neq \mathbf{x}$; in an infinitesimal time interval $\delta t$, the quantity $f(\mathbf{x}'|\mathbf{x})\,\delta t$ represents the infinitesimal probability that the system will make a transition from state $\mathbf{x}$ at time $t$ to state $\mathbf{x}'$ at time $t + \delta t$. Explicitly,

$$p(\mathbf{x}'|\mathbf{x}) \simeq \delta_{\mathbf{x}'\mathbf{x}} + \delta t f(\mathbf{x}'|\mathbf{x}) \tag{1}$$

where $\delta_{\mathbf{x}'\mathbf{x}}$ is the Kronecker delta and the equation becomes exact in the limit $\delta t \to 0$. Equation (1) implies by normalisation that $f(\mathbf{x}|\mathbf{x}) = -\sum_{\mathbf{x}' \neq \mathbf{x}} f(\mathbf{x}'|\mathbf{x})$. The interpretation of the process rates as infinitesimal transition probabilities highlights the simple relationship between the marginal distribution $p_t(\mathbf{x})$ and the process rates. The probability of finding the system in state $\mathbf{x}$ at time $t + \delta t$ will be given by the probability that the system was already in state $\mathbf{x}$ at time $t$, minus the probability that the system was in state $\mathbf{x}$ at time $t$ and jumped to state $\mathbf{x}'$, plus the probability that the system was in a different state $\mathbf{x}''$ at time $t$ and then jumped to state $\mathbf{x}$. In formulae, this is given by

$$p_{t+\delta t}(\mathbf{x}) = p_t(\mathbf{x}) \left[ 1 - \sum_{\mathbf{x}' \neq \mathbf{x}} f(\mathbf{x}'|\mathbf{x})\,\delta t \right] + \sum_{\mathbf{x}' \neq \mathbf{x}} p_t(\mathbf{x}')\,f(\mathbf{x}|\mathbf{x}')\,\delta t.$$

Taking the limit for $\delta t \to 0$ we obtain the (forward) *Master equation* for the marginal probabilities

$$\frac{dp_t(\mathbf{x})}{dt} = \sum_{\mathbf{x}' \neq \mathbf{x}} \left[ -p_t(\mathbf{x})\,f(\mathbf{x}'|\mathbf{x}) + p_t(\mathbf{x}')\,f(\mathbf{x}|\mathbf{x}') \right]. \tag{2}$$

# 2 Variational approximate inference

Let us assume that we have noisy observations $\mathbf{y}_l \quad l = 1, \ldots, N$ of the state of the system at a discrete number of time points; the noise model is specified by a likelihood function $\hat{p}(\mathbf{y}_l|\mathbf{x}(t_l))$. We can combine this likelihood with the prior process to obtain a posterior process. As the observations happen at discrete time points, the posterior process is clearly still a Markov jump process. Given the Markovian nature of the processes, one could hope to obtain the posterior rate functions $g(\mathbf{x}'|\mathbf{x})$ by a forward-backward procedure similar to the one used for Hidden Markov Models. While this is possible in principle, the computations would require simultaneously solving a very large system of coupled linear ODEs (the number of equations is of order $S^D$, $S$ being the number of states accessible to the system), which is not feasible even in simple systems.

In the following, we will use the variational mean field (MF) approach to approximate the posterior process by a factorizing process, minimising the *Kullback - Leibler* (KL) divergence between processes. The inference process is then reduced to the solution of $D$ *one - dimensional* Master and backward equations of size $S$. This is still nontrivial because the KL divergence requires the joint probabilities of variables $\mathbf{x}(t)$ at infinitely many different times $t$, i.e. probabilities over entire paths of a process rather than the simpler marginals $p_t(\mathbf{x})$. We will circumvent this problem by working with time discretised trajectories and then passing on to the continuum time limit. We denote such a trajectory as $\mathbf{x}_{0:K} = (\mathbf{x}(t_0), \ldots, \mathbf{x}(t_0 + K\delta t))$ where $\delta t$ is a small time interval and $K$ is very large. Hence, we write the joint posterior probability as

$$p_{post}(\mathbf{x}_{0:K}) = \frac{1}{Z} p_{prior}(\mathbf{x}_{0:K}) \times \prod_{l=1}^{N} \hat{p}(\mathbf{y}_l|\mathbf{x}(t_l)) \quad \text{with} \quad p_{prior}(\mathbf{x}_{0:K}) = p(\mathbf{x}_0) \prod_{k=0}^{K-1} p(\mathbf{x}_{k+1}|\mathbf{x}_k)$$

with $Z = p(\mathbf{y}_1, \ldots, \mathbf{y}_N)$. Note that $\mathbf{x}(t_l) \in \mathbf{x}_{0:K}$. In the rest of this section, we will show how to compute the posterior rates and marginals by minimising the KL divergence. We notice in passing that a similar framework for continuous stochastic processes was proposed recently in [8].

## 2.1 KL divergence between MJPs

The KL divergence between two MJPs defined by their path probabilities $p(\mathbf{x}_{0:K})$ and $q(\mathbf{x}_{0:K})$ is

$$KL[q,p] = \sum_{\mathbf{x}_{0:K}} q(\mathbf{x}_{0:K}) \ln \frac{q(\mathbf{x}_{0:K})}{p(\mathbf{x}_{0:K})} = \sum_{k=0}^{K-1} \sum_{\mathbf{x}_k} q(\mathbf{x}_k) \sum_{\mathbf{x}_{k+1}} q(\mathbf{x}_{k+1}|\mathbf{x}_k) \ln \frac{q(\mathbf{x}_{k+1}|\mathbf{x}_k)}{p(\mathbf{x}_{k+1}|\mathbf{x}_k)} + K_0$$

and where $K_0 = \sum_{\mathbf{x}_0} q(\mathbf{x}_0) \log \frac{q(\mathbf{x}_0)}{p(\mathbf{x}_0)}$ will be set to zero in the following. We can now use equation (1) for the conditional probabilities; letting $\delta t \to 0$ and simultaneously $K \to \infty$ so that $K\delta t \to T$, we obtain

$$KL[q,p] = \int_0^T dt \sum_{\mathbf{x}} q_t(\mathbf{x}) \sum_{\mathbf{x}':\mathbf{x}'\neq\mathbf{x}} \left\{ g(\mathbf{x}'|\mathbf{x}) \ln \frac{g(\mathbf{x}'|\mathbf{x})}{f(\mathbf{x}'|\mathbf{x})} + f(\mathbf{x}'|\mathbf{x}) - g(\mathbf{x}'|\mathbf{x}) \right\} \qquad (3)$$

where $f(\mathbf{x}'|\mathbf{x})$ and $g(\mathbf{x}'|\mathbf{x})$ are the rates of the $p$ and $q$ process respectively. Notice that we have swapped from the path probabilities framework to an expression that depends solely on the process rates and marginals.

## 2.2 MF approximation to posterior MJPs

We will now consider the case where $p$ is a posterior MJP and $q$ is an approximating process. The prior process will be denoted as $p_{prior}$ and its rates will be denoted by $f$. The KL divergence then is

$$KL(q, p_{post}) = \ln Z + KL(q, p_{prior}) - \sum_{l=1}^{N} E_q[\ln \hat{p}(\mathbf{y}_l|\mathbf{x}(t_l))].$$

To obtain a tractable inference problem, we will assume that, in the approximating process $q$, *the joint path probability for all the species factorises into the product of path probabilities for individual species*. This gives the following equations for the species probabilities and transition rates

$$q_t(\mathbf{x}) = \prod_{i=1}^{D} q_{it}(x_i) \qquad g_t(\mathbf{x}'|\mathbf{x}) = \sum_{i=1}^{D} \prod_{j\neq i} \delta_{x'_j,x_j} g_{it}(x'_i|x_i). \qquad (4)$$

Notice that we have emphasised that the process rates for the approximating process may depend explicitly on time, even if the process rates of the original process do not. Exploiting these assumptions, we obtain that the KL divergence between the approximating process and the posterior process is given by

$$KL[q, p_{post}] = \ln Z - \sum_{l=1}^{N} E_q[\ln \hat{p}(\mathbf{y}_l|\mathbf{x}(t_l))] +$$

$$\int_0^T dt \sum_i \sum_x q_{it}(x) \sum_{x':x'\neq x} \left\{ g_{it}(x'|x) \ln \frac{g_{it}(x'|x)}{\hat{f}_i(x'|x)} + \tilde{f}_i(x'|x) - g_{it}(x'|x) \right\} \qquad (5)$$

where we have defined

$$\hat{f}_i\left(x'|x\right) = \exp\left(E_{\mathbf{x}\backslash i}[\ln f_i\left(\mathbf{x}'|\mathbf{x} : x_j' = x_j, \forall j \neq i\right)]\right)$$
$$\tilde{f}_i\left(x'|x\right) = E_{\mathbf{x}\backslash i}[f_i\left(\mathbf{x}'|\mathbf{x} : x_j' = x_j, \forall j \neq i\right)] \tag{6}$$

and $E_{\mathbf{x}\backslash i}[\ldots]$ denotes an expectation over all components of $\mathbf{x}$ except $x_i$ (using the measure $q$). In order to find the MF approximation to the posterior process we must optimise the KL divergence (5) with respect to the marginals $q_{it}(x)$ and the rates $g_{it}\left(x'|x\right)$. These, however, are not independent but fulfill the Master equation (2).

We will take care of this constraint by using a Lagrange multiplier function $\lambda_i(x,t)$ and compute the stationary values of the Lagrangian

$$L = KL\left(q, p_{post}\right)$$
$$- \sum_i \int_0^T dt \sum_x \lambda_i\left(x,t\right)\left(\partial_t q_{it}\left(x\right) - \sum_{x' \neq x}\left\{g_{it}\left(x|x'\right)q_{it}\left(x'\right) - g_{it}\left(x'|x\right)q_{it}\left(x\right)\right\}\right). \tag{7}$$

We can now compute functional derivatives of (7) to obtain

$$\frac{\delta L}{\delta q_{it}(x)} = \sum_{x' \neq x}\left[g_{it}\left(x'|x\right)\ln\frac{g_{it}\left(x'|x\right)}{\hat{f}_i\left(x'|x\right)} - g_{it}\left(x'|x\right) + \tilde{f}_i\left(x'|x\right)\right] + \partial_t \lambda_i\left(x,t\right) +$$
$$\sum_{x'} g_{it}\left(x'|x\right)\left\{\lambda_i\left(x',t\right) - \lambda_i\left(x,t\right)\right\} - \sum_l \ln\hat{p}\left(\mathbf{y}_l|\mathbf{x}\left(t\right)\right)\delta\left(t - t_l\right) = 0 \tag{8}$$

$$\frac{\delta L}{\delta g_{it}\left(x'|x\right)} = q_{it}\left(x\right)\left(\ln\frac{g_{it}\left(x'|x\right)}{\hat{f}_i\left(x'|x\right)} + \lambda_i\left(x',t\right) - \lambda_i\left(x,t\right)\right) = 0 \tag{9}$$

Defining $r_i(x,t) = e^{-\lambda_i(x,t)}$ and inserting (9) into (8), we arrive at the linear differential equation

$$\frac{dr_i(x,t)}{dt} = \sum_{x' \neq x}\left(\tilde{f}_i\left(x'|x\right)r_i\left(x,t\right) - \hat{f}_i\left(x'|x\right)r_i\left(x',t\right)\right) \tag{10}$$

valid for all times outside of the observations. To include the observations, we assume for simplicity that the noise model factorises across the species, so that $\hat{p}\left(\mathbf{y}_l|\mathbf{x}(t)\right) = \prod_i \hat{p}_i\left(y_{il}|x_i(t_l)\right) \quad \forall l$. Then equation (8) yields

$$\lim_{t \to t_l^-} r_i\left(x,t\right) = \hat{p}_i\left(y_{il}|x_i(t_l)\right)\lim_{t \to t_l^+} r_i\left(x,t\right).$$

We can then optimise the Lagrangian (7) using an iterative strategy. Starting with an initial guess for $q_t(x)$ and selecting a species $i$, we can compute $\hat{f}_i\left(x'|x\right)$ and $\tilde{f}_i\left(x'|x\right)$. Using these, we can solve equation (10) backwards starting from the condition $r_i\left(x,T\right) = 1 \forall x$ (i.e., the constraint becomes void at the end of the time under consideration). This allows us to update our estimate of the rates $g_{it}\left(x'|x\right)$ using equation (9), which can then be used to solve the master equation (2) and update our guess of $q_{it}(x)$. This procedure can be followed sequentially for all the species; as each step leads to a decrease in the value of the Lagrangian, this guarantees that the algorithm will converge to a (local) minimum.

## 2.3 Parameter estimation

Since $KL\left[q, p_{post}\right] \geq 0$, we obtain as useful by-product of the MF approximation a tractable variational lower bound on the log - likelihood of the data $\log Z = \log p(\mathbf{y}_1, \ldots, \mathbf{y}_N)$ from (5). As usual [e.g 7] such a bound can be used in order to optimise model parameters using a variational E-M algorithm.

# 3 Example: the Lotka-Volterra process

The Lotka-Volterra (LV) process is often used as perhaps the simplest non-trivial MJP [6, 4]. Introduced independently by Alfred J. Lotka in 1925 and Vito Volterra in 1926, it describe the dynamics of a population composed of two interacting species, traditionally referred to as predator and prey. The process rates for the LV system are given by

$$f_{prey}\,(x+1|x,y) = \alpha x \qquad\qquad f_{prey}\,(x-1|x,y) = \beta xy$$
$$f_{predator}\,(y+1|x,y) = \delta xy \qquad\qquad f_{prey}\,(y-1|x,y) = \gamma y \qquad (11)$$

where $x$ is the number of preys and $y$ is the number of predators. All other rates are zero: individuals can only be created or destroyed one at the time. Rate sparsity is a characteristic of very many processes, including all chemical kinetic processes (indeed, the LV model can be interpreted as a chemical kinetic model). An immediate difficulty in implementing our strategy is that some of the process rates are identically zero when one of the species is extinct (*i.e.* its numbers have reached zero); this will lead to infinities when computing the expectation of the logarithm of the rates in equation (6). To avoid this, we will "regularise" the process by adding a small constant to the $f(1|0)$; it can be proved that on average over the data generating process the variational approximation to the regularised process still optimises a bound analogous to (3) on the original process [9].

The variational estimates for the parameters of the LV process are obtained by inserting the process rates (11) into the MF bound and taking derivatives w.r.t. the parameters. Setting them to zero, we obtain a set of fixed point equations

$$\alpha = \frac{\int_0^T \langle g_{preyt}\,(x+1|x)\rangle_{preyt}}{\int_0^T dt\,\langle x\rangle_{preyt}}, \qquad\qquad \beta = \frac{\int_0^T \langle g_{preyt}\,(x-1|x)\rangle_{preyt}}{\int_0^T dt\,\langle x\rangle_{preyt}\,\langle y\rangle_{predatort}},$$

$$\gamma = \frac{\int_0^T \langle g_{predatort}\,(y-1|y)\rangle_{predatort}}{\int_0^T dt\,\langle y\rangle_{predatort}}, \qquad\qquad \delta = \frac{\int_0^T \langle g_{predatort}\,(y+1|y)\rangle_{predatort}}{\int_0^T dt\,\langle y\rangle_{predatort}\,\langle x\rangle_{preyt}}. \qquad (12)$$

Equations (12) have an appealing intuitive meaning in terms of the physics of the process: for example, $\alpha$ is given by the average total increase rate of the approximating process divided by the average total number of preys.

We generated 15 counts of predator and prey numbers at regular intervals from a LV process with parameters $\alpha = 5 \times 10^{-4}$, $\beta = 1 \times 10^{-4}$, $\gamma = 5 \times 10^{-4}$ and $\delta = 1 \times 10^{-4}$, starting from initial population levels of seven predators and nineteen preys. These counts were then corrupted according to the following noise model

$$\hat{p}_i\,(y_{il}|x_i\,(t_l)) \propto \left[ \frac{1}{2^{|y_{il}-x_i(t_l)|}} + 10^{-6} \right], \qquad (13)$$

where $x_i\,(t_l)$ is the (discrete) count for species $i$ at time $t_l$ before the addition of noise. Notice that, since population numbers are constrained to be positive, the noise model is not symmetric. The original count is placed at the mode, rather than the mean, of the noise model. This asymmetry is unavoidable when dealing with quantities that are constrained positive.

While in theory each species can have an arbitrarily large number of individuals, in order to solve the differential equations (2) and (10) we have to truncate the process. While the truncation threshold could be viewed as another parameter and optimised variationally, in these experiments we took a more heuristic approach and limited the maximum number of individuals of each species to 200. This was justified by considering that an exponential growth pattern fitted to the available data led to an estimate of approximately 90 individuals in the most abundant species, well below the truncation threshold.

The results of the inference are shown in Figure 1. The solid line is the mean of the approximating distribution, the dashed lines are the 90% confidence intervals, the dotted line is the true path from which the data was obtained. The diamonds are the noisy observations. The parameter values inferred are reasonably close to the real parameter values: $\alpha = 1.35 \times 10^{-3}$, $\beta = 2.32 \times 10^{-4}$,

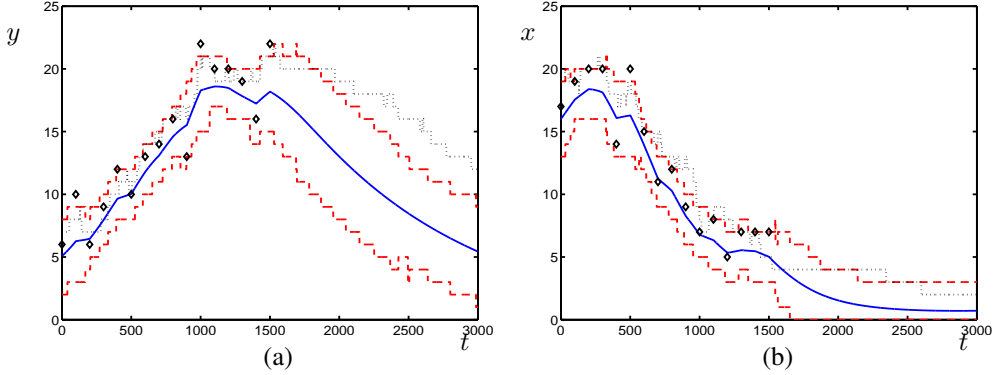

Figure 1: MF approximation to posterior LV process: (a) predator population and (b) prey population. Diamonds are the (noisy) observed data points, solid line the mean, dashed lines 90% confidence intervals, dotted lines the true path from which the data was sampled.

$\gamma = 1.57 \times 10^{-3}$ and $\delta = 1.78 \times 10^{-4}$. While the process is well approximated in the area where data is present, the free-form prediction is less good, especially for the predator population. This might be due to the inaccuracies in the estimates of the parameters. The approximate posterior displays nontrivial emerging properties: for example, we predict that there is a 10% chance that the prey population will become extinct at the end of the period of interest. These results were obtained in approximately fifteen minutes on an Intel Pentium M 1.7GHz laptop computer.

To check the reliability of our inference results and the rate with which the estimated parameter values converge to the true values, we repeated our experiments for 5, 10, 15 and 20 available data points. For each sample size, we drew five independent samples from the same LV process. Figure 2(a) shows the average and standard deviation of the mean squared error (MSE) in the estimate of the parameters as a function of the number of observations $N$; as expected, this decreases uniformly with the sample size.

## 4   Example: gene autoregulatory network

As a second example we consider a gene autoregulatory network. This simple network motif is one of the most important building blocks of the transcriptional regulatory networks found in cells because of its ability to increase robustness in the face of fluctuation in external signals [10]. Because of this, it is one of the best studied systems, both at the experimental and at the modelling level [11, 3]. The system consists again of two species, mRNA and protein; the process rates are given by

$$f_{RNA}(x + 1|x, y) = \alpha \left(1 - 0.99 \times \Theta \left(y - y_c\right)\right) \qquad f_{RNA}\left(x - 1|x, y\right) = \beta x$$
$$f_p\left(y + 1|x, y\right) = \gamma x \qquad\qquad f_p\left(y - 1|x, y\right) = \delta y \qquad (14)$$

where $\Theta$ is the Heavyside step function, $y$ the protein number and $x$ the mRNA number. The intuitive meaning of these equations is simple: both protein and mRNA decay exponentially. Proteins are produced through *translation* of mRNA with a rate proportional to the mRNA abundance. On the other hand, mRNA production depends on protein concentration levels through a logical function: as soon as protein numbers increase beyond a certain critical parameter $y_c$, mRNA production drops dramatically by a factor 100.

The optimisation of the variational bound w.r.t. the parameters $\alpha$, $\beta$, $\gamma$ and $\delta$ is straightforward and yields fixed point equations similar to the ones for the LV process. The dependence of the MF bound on the critical parameter $y_c$ is less straightforward and is given by

$$\mathcal{L}_{y_c} = const + \left\{ 2 \int_0^T dt \bar{g} h\left(y_c\right) + \log \left[1 - 0.99 \frac{1}{T} \int_0^T h\left(y_c\right) dt\right] \int_0^T dt \bar{g} \right\} \qquad (15)$$

where $\bar{g} = \langle g_{RNA}\left(x + 1|x\right)\rangle_{q_{RNA}}$ and $h\left(y_c\right) = \sum_{y \geq y_c} q_p\left(y\right)$. A plot of this function obtained during the inference task below can be seen in Figure 2(b). We can determine the minimum of (15) by searching over the possible (discrete) values of $y_c$.

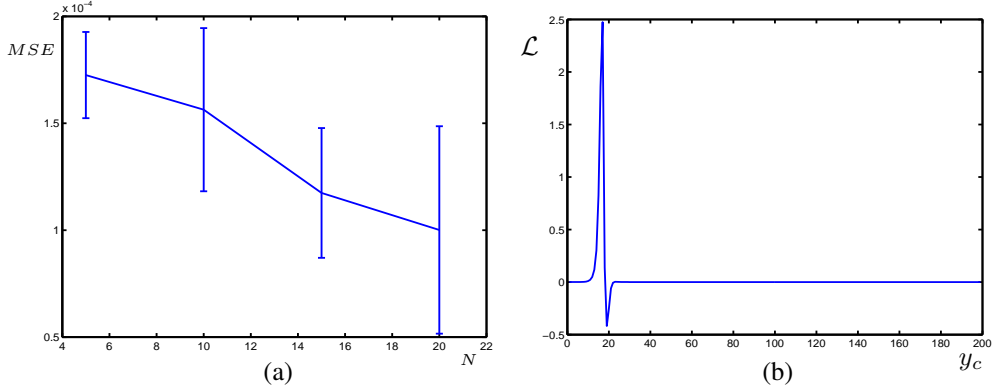

Figure 2: (a) Mean squared error (MSE) in the estimate of the parameters as a function of the number of observations $N$ for the LV process. (b) Negative variational likelihood bound for the gene autoregulatory network as a function of the critical parameter $y_c$.

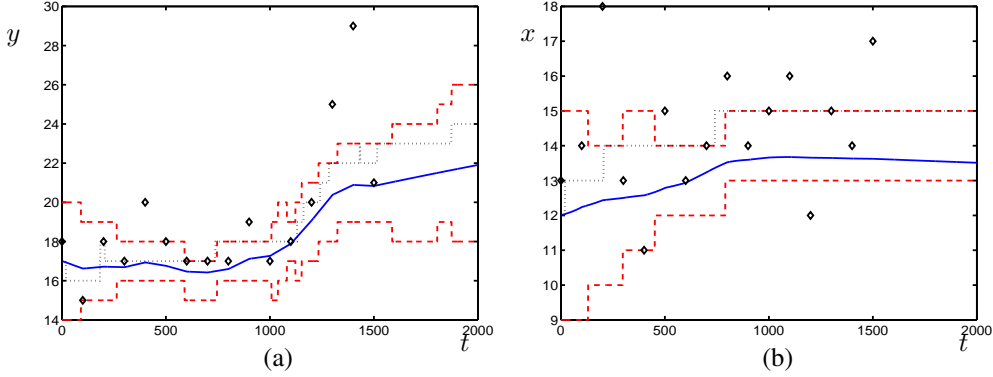

Figure 3: MF approximation to posterior autoregulatory network process: (a) protein population and (b) mRNA population. Diamonds are the (noisy) observed data points, solid line the mean, dashed lines 90% confidence intervals, dotted lines the true path from which the data was taken.

Again, we generated data by simulating the process with parameter values $y_c = 20$, $\alpha = 2 \times 10^{-3}$, $\beta = 6 \times 10^{-5}$, $\gamma = 5 \times 10^{-4}$ and $\delta = 7 \times 10^{-5}$. Fifteen counts were generated for both mRNA and proteins, with initial count of 17 protein and 12 mRNA molecules. These were then corrupted with noise generated from the distribution shown in equation (13). The results of the approximate posterior inference are shown in Figure 3. The inferred parameter values are in good agreement with the true values: $y_c = 19$, $\alpha = 2.20 \times 10^{-3}$, $\beta = 1.84 \times 10^{-5}$, $\gamma = 4.01 \times 10^{-4}$ and $\delta = 1.54 \times 10^{-4}$. Interestingly, if the data is such that the protein count never exceeds the critical parameter $y_c$, this becomes unidentifiable (the likelihood bound is optimised by $y_c = \infty$ or $y_c = 0$), as may be expected. The likelihood bound loses its sharp optimum evident from Figure 2(b) (results not shown).

## 5   Discussion

In this contribution we have shown how a MF approximation can be used to perform posterior inference in MJPs from discretely observed noisy data. The MF approximation has been shown to perform well and to retain much of the richness of these complex systems. The proposed approach is conceptually very different from existing MCMC approaches [6]. While these focus on sampling from the distribution of reactions happening in a small interval in time, we compute an approximation to the probability distribution over possible paths of the system. This allows us to easily factorise across species; by contrast, sampling the number of reactions happening in a certain time

interval is difficult, and not amenable to simple techniques such as Gibbs sampling. While it is possible that future developments will lead to more efficient sampling strategies, our approach outstrips current MCMC based methods in terms of computational efficiency, A further strength of our approach is the ease with which it can be scaled to more complex systems involving larger numbers of species. The factorisation assumption implies that the computational complexity grows linearly in the number of species $D$; it is unclear how MCMC would scale to larger systems.

An alternative suggestion, proposed in [11], was somehow to seek a middle way between a MJP and a deterministic, ODE based approach by approximating the MJP with a continuous *stochastic* process, *i.e.* by using a diffusion approximation. While these authors show that this approximation works reasonably well for inference purposes, it is worth pointing out that the population sizes in their experimental results were approximately one order of magnitude larger than in ours. It is arguable that a diffusion approximation might be suitable for population sizes as low as a few hundreds, but it cannot be expected to be reasonable for population sizes of the order of 10.

The availability of a practical tool for statistical inference in MJPs opens a number of important possible developments for modelling. It would be of interest, for example, to develop mixed models where one species with low counts interacts with another species with high counts that can be modelled using a deterministic or diffusion approximation. This situation would be of particular importance for biological applications, where different proteins can have very different copy numbers in a cell but still be equally important. Another interesting extension is the possibility of introducing a spatial dimension which influences how likely interactions are. Such an extension would be very important, for example, in an epidemiological study. All of these extensions rely centrally on the possibility of estimating posterior probabilities, and we expect that the availability of a practical tool for the inference task will be very useful to facilitate this.

## References

[1] Harley H. McAdams and Adam Arkin. Stochastic mechanisms in gene expression. *Proceedings of the National Academy of Sciences USA*, 94:814–819, 1997.

[2] Long Cai, Nir Friedman, and X. Sunney Xie. Stochastic protein expression in individual cells at the single molecule level. *Nature*, 440:580–586, 2006.

[3] Yoshito Masamizu, Toshiyuki Ohtsuka, Yoshiki Takashima, Hiroki Nagahara, Yoshiko Takenaka, Kenichi Yoshikawa, Hitoshi Okamura, and Ryoichiro Kageyama. Real-time imaging of the somite segmentation clock: revelation of unstable oscillators in the individual presomitic mesoderm cells. *Proceedings of the National Academy of Sciences USA*, 103:1313–1318, 2006.

[4] Daniel T. Gillespie. Exact stochastic simulation of coupled chemical reactions. *Journal of Physical Chemistry*, 81(25):2340–2361, 1977.

[5] Eric Mjolsness and Guy Yosiphon. Stochastic process semantics for dynamical grammars. to appear in Annals of Mathematics and Artificial Intelligence, 2006.

[6] Richard J. Boys, Darren J. Wilkinson, and Thomas B. L. Kirkwood. Bayesian inference for a discretely observed stochastic kinetic model. available from http://www.staff.ncl.ac.uk/d.j.wilkinson/pub.html, 2004.

[7] Manfred Opper and David Saad (editors). *Advanced Mean Field Methods*. MIT press, Cambridge,MA, 2001.

[8] Cedric Archambeau, Dan Cornford, Manfred Opper, and John Shawe-Taylor. Gaussian process approximations of stochastic differential equations. *Journal of Machine Learning Research Workshop and Conference Proceedings*, 1(1):1–16, 2007.

[9] Manfred Opper and David Haussler. Bounds for predictive errors in the statistical mechanics of supervised learning. *Physical Review Letters*, 75:3772–3775, 1995.

[10] Uri Alon. *An introduction to systems biology*. Chapman and Hall, London, 2006.

[11] Andrew Golightly and Darren J. Wilkinson. Bayesian inference for stochastic kinetic models using a diffusion approximation. *Biometrics*, 61(3):781–788, 2005.

